# Augmented-SVM: Automatic space partitioning for combining multiple non-linear dynamics

**Ashwini Shukla**
ashwini.shukla@epfl.ch

**Aude Billard**
aude.billard@epfl.ch

Learning Algorithms and Systems Laboratory (LASA)
École Polytechnique Fédérale de Lausanne (EPFL)
Lausanne, Switzerland - 1015

## Abstract

Non-linear dynamical systems (DS) have been used extensively for building generative models of human behavior. Their applications range from modeling brain dynamics to encoding motor commands. Many schemes have been proposed for encoding robot motions using dynamical systems with a single attractor placed at a predefined target in state space. Although these enable the robots to react against sudden perturbations without any re-planning, the motions are always directed towards a single target. In this work, we focus on combining several such DS with distinct attractors, resulting in a multi-stable DS. We show its applicability in reach-to-grasp tasks where the attractors represent several grasping points on the target object. While exploiting multiple attractors provides more flexibility in recovering from unseen perturbations, it also increases the complexity of the underlying learning problem. Here we present the *Augmented-SVM* (A-SVM) model which inherits region partitioning ability of the well known SVM classifier and is augmented with novel constraints derived from the individual DS. The new constraints modify the original SVM dual whose optimal solution then results in a new class of support vectors (SV). These new SV ensure that the resulting multi-stable DS incurs minimum deviation from the original dynamics and is stable at each of the attractors within a finite region of attraction. We show, via implementations on a simulated 10 degrees of freedom mobile robotic platform, that the model is capable of real-time motion generation and is able to adapt on-the-fly to perturbations.

## 1 Introduction

Dynamical systems (DS) have proved to be a promising framework for encoding and generating complex motions. A major advantage of representing motion using DS based models [1, 2, 3, 4] is the ability to counter perturbations by virtue of the fact that re-planning of trajectories is instantaneous. These are generative schemes that define the flow of trajectories in state space $\mathbf{x} \in \mathbb{R}^N$ by means of a non-linear dynamical function $\dot{\mathbf{x}} = f(\mathbf{x})$. DS with single stable attractors have been used in pick and place tasks to control for both the motion of the end-effector [5, 6, 7] and the placement of the fingers on an object [8]. Assuming a single attractor, and hence a single grasping location on the object, constrains considerably the applicability of these methods to realistic grasping problems. A DS composed of multiple stable attractors provides an opportunity to encode different ways to reach and grasp an object. Recent neuro-physiological results [9] have shown that a DS based modeling best explains the trajectories followed by humans while switching between several reaching targets. From a robotics viewpoint, a robot controlled using a DS with multiple attractors would

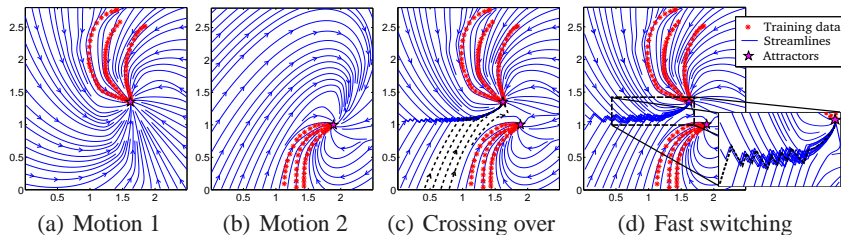

(a) Motion 1     (b) Motion 2     (c) Crossing over     (d) Fast switching

Figure 2: Combining motions using naive SVM classification based switching.

be able to switch online across grasping strategies. This may be useful, e.g., when one grasping point becomes no longer accessible due to a sudden change in the orientation of the object or the appearance of an obstacle along the current trajectory. This paper presents a method by which one can learn multiple dynamics directed toward different attractors in a single dynamical system.

The dynamical function $f(\mathbf{x})$ is usually estimated using non-linear regression functions such as Gaussian Process Regression (GPR) [10], Gaussian Mixture Regression (GMR) [7], Locally Weighted Projection Regression (LWPR) [11] or Dynamical Movement Primitives (DMP) [1]. However, all of these works modeled DS with a single attractor. While [7, 12] ensure global stability at the attractor, other approaches result in unstable DS with spurious attractors.

Stability at multiple targets has been addressed to date largely through neural networks approaches. The Hopfield network and variants offered a powerful means to encode several stable attractors in the same system to provide a form of content-addressable memory [13, 14]. The

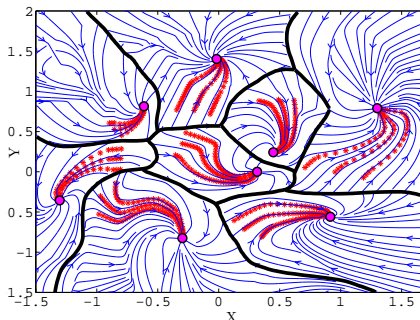

Figure 1: 8 attractor DS

dynamics to reach these attractors was however not controlled for, nor was the partitioning of the state space that would send the trajectories to each attractor. Echo-state networks provide alternative ways to encode various complex dynamics [15]. Although they have proved to be universal estimators, their ability to generalize in untrained regions of state space remains unverified. Also, the key issue of global stability of the learned dynamics is achieved using heuristic rules. To our knowledge, this is the first attempt at learning simultaneously a partitioning of the state space and an embedding of multiple dynamical systems with separate regions of attractions and distinct attractors.

## 2 Preliminaries

A naive approach to building a multi-attractor DS would be to first partition the space and then learn a DS in each partition separately. This would unfortunately rarely result in the desired compound system. Consider, for instance, two DS with distinct attractors, as shown in Fig. 2(a)-(b). First, we build a SVM classifier to separate data points of the first DS, labeled $+1$, from data points of the other DS, labeled $-1$. We then estimate each DS separately using any of the techniques reviewed in the previous section. Let $h : \mathbb{R}^N \mapsto \mathbb{R}$ denote the classifier function that separates the state space $\mathbf{x} \in \mathbb{R}^N$ into two regions with labels $y_i \in \{+1, -1\}$. Also, let the two DS be $\dot{\mathbf{x}} = f_{y_i}(\mathbf{x})$ with stable attractors at $\mathbf{x}^*_{y_i}$. The combined DS is then given by $\dot{\mathbf{x}} = f_{\mathrm{sgn}(h(\mathbf{x}))}(\mathbf{x})$. Figure 2(c) shows the trajectories resulting from this approach. Due to the non-linearity of the dynamics, trajectories initialized in one region cross the boundary and converge to the attractor located in the opposite region. In other words, each region partitioned by the SVM hyperplane is not a region of attraction for its attractor. In a real-world scenario where the attractors represent grasping points on an object and the trajectories are to be followed by robots, crossing over may take the trajectories towards kinematically unreachable regions. Also, as shown in Fig. 2(d), trajectories that encounter the boundary may switch rapidly between different dynamics leading to jittery motion.

To ensure that the trajectories do not cross the boundary and remain within the region of attraction of their respective attractors, one could adopt a more informed approach in which each of the

original DS is modulated such that the generated trajectories always move away from the classifier boundary. Recall that by construction, the absolute value of the classifier function $h(\mathbf{x})$ increases as one moves away from the classification hyperplane. The gradient $\nabla h(\mathbf{x})$ is hence positive, respectively negative, as one moves inside the region of the positive, respectively negative, class. We can exploit this observation to deflect selective components of the velocity signal from the original DS along, respectively opposite to, the direction $\nabla h(\mathbf{x})$. Concretely, if $\dot{\mathbf{x}}_O = f_{\text{sgn}(h(\mathbf{x}))}(\mathbf{x})$ denotes the velocity obtained from the original DS and

$$\lambda(\mathbf{x}) = \begin{cases} \max\left(\epsilon, \nabla h(\mathbf{x})^T \dot{\mathbf{x}}_O\right) & \text{if } h(\mathbf{x}) > 0 \\ \min\left(-\epsilon, \nabla h(\mathbf{x})^T \dot{\mathbf{x}}_O\right) & \text{if } h(\mathbf{x}) < 0 \end{cases}, \quad (1)$$

the modulated dynamical system is given by

$$\dot{\mathbf{x}} = \tilde{f}(\mathbf{x}) = \lambda(\mathbf{x})\nabla h(\mathbf{x}) + \dot{\mathbf{x}}_\perp. \quad (2)$$

Here, $\epsilon$ is a small positive scalar and $\dot{\mathbf{x}}_\perp = \dot{\mathbf{x}}_O - \left(\frac{\nabla h(\mathbf{x})^T \dot{\mathbf{x}}_O}{\|\nabla h(\mathbf{x}\|^2}\right)\nabla h(\mathbf{x})$ is the component of the original velocity perpendicular to $\nabla h$. This results in a vector field that flows along increasing values of the classifier function in the regions of space where $h(\mathbf{x}) > 0$ and along decreasing values for $h(\mathbf{x}) < 0$. As a result, the trajectories move away from the classification hyperplane and converge to a point located in the region where they were initialized. Such modulated systems have been used extensively for estimating stability regions of interconnected power networks [16] and are known as *quasi gradient systems* [17]. If $h(\mathbf{x})$ is upper bounded[1], all trajectories converge to one of the stationary points $\{\mathbf{x} : \nabla h(\mathbf{x}) = 0\}$ and $h(\mathbf{x})$ is a Lyapunov function of the overall system (refer [17], proposition 1).

Figure 3 shows the result of applying the above modulation to our pair of DS. As expected, it forces the trajectories to flow along the gradient of the function $h(\mathbf{x})$. Although this solves the problem of "crossing-over" the boundary, the trajectories obtained are deficient in two major ways. They depart heavily from the original dynamics and do not terminate at the desired attractors. This is due to the fact that the function $h(\mathbf{x})$ used to modulate the DS was designed solely for classification and contained no information about the dynamics of the two original DS. In other words, the vector field given by $\nabla h(\mathbf{x})$ was not aligned with the flow of the training trajectories and the stationary points of the modulation function did not coincide with the desired attractors.

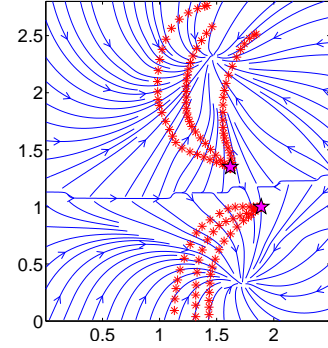

Figure 3: Modulated trajs.

In subsequent sections, we show how we can learn a new modulation function which takes into account the three issues we highlighted in this preliminary discussion. We will seek a system that $a$) ensures strict classification across regions of attraction (ROA) for each DS, $b$) follows closely the dynamics of each DS in each ROA and $c$) ensures that all trajectories in each ROA reach the desired attractor. Satisfying requirements $a$) and $b$) above is equivalent to performing classification and regression simultaneously. We take advantage of the fact that the optimization in support vector classification and support vector regression have the same form to phrase our problem in a single constrained optimization framework. In next sections, we show that in addition to the usual SVM support vectors (SVs), the resulting modulation function is composed of an additional class of SVs. We geometrically analyze the effect of these new support vectors on the resulting dynamics. While this preliminary discussion considered solely binary classification, we will now extend the problem to multi-class classification.

## 3  Problem Formulation

The $N$-dimensional state space of the system represented by $\mathbf{x} \in \mathbb{R}^N$ is partitioned into $M$ different classes, one for each of the $M$ motions to be combined. We collect trajectories in the state space, yielding a set of $P$ data points $\{\mathbf{x}_i; \dot{\mathbf{x}}_i; l_i\}_{i=1...P}$ where $l_i \in \{1, 2, \cdots, M\}$ refers to the class label of each point[2]. To learn the set of modulation functions $\{h_m(\mathbf{x})\}_{m=1...M}$, we proceed recursively. We learn each modulation function in a one-vs-all classifier scheme and then

compute the final modulation function $\tilde{h}(\mathbf{x}) = \max\limits_{m=1\cdots M} h_m(\mathbf{x})$. In the multi-class setting, the behavior of avoiding boundaries is obtained if the trajectories move along *increasing* values of the function $\tilde{h}(\mathbf{x})$. To this effect, the deflection term $\lambda(\mathbf{x})$ presented in the binary case 1 becomes $\lambda(\mathbf{x}) = \max\left(\epsilon, \nabla\tilde{h}(\mathbf{x})^T\dot{\mathbf{x}}_O\right); \forall\mathbf{x} \in \mathbb{R}^N$. Next, we describe the procedure for learning a single $h_m(\mathbf{x})$ function.

We follow classical SVM formulation and lift the data into a higher dimensional feature space through the mapping $\phi : \mathbb{R}^N \mapsto \mathbb{R}^F$ where $F$ denotes the dimension of the feature space. We also assume that each function $h_m(\mathbf{x})$ is linear in feature space, i.e., $h_m(\mathbf{x}) = \mathbf{w}^T\phi(\mathbf{x}) + b$ where $\mathbf{w} \in \mathbb{R}^F, b \in \mathbb{R}$. We label the current ($m-th$) motion class as positive and all others negative such that the set of labels for the current sub-problem is given by

$$y_i = \begin{cases} +1 & \text{if } l_i = m \\ -1 & \text{if } l_i \neq m \end{cases} ; \qquad i = 1\cdots P.$$

Also, the set indexing the positive class is then defined as $\mathcal{I}_+ = \{i : i \in [1, P]; l_i = m\}$. With this, we formalize the three constraints explained in Section 2 as:

**Region separation:** Each point must be classified correctly yields $P$ constraints:

$$y_i\left(\mathbf{w}^T\phi(\mathbf{x}_i) + b\right) \geq 1 \quad \forall i = 1...P. \tag{3}$$

**Lyapunov constraint:** To ensure that the modulated flow is aligned with the training trajectories, the gradient of the modulation function must have a positive component along the velocities at the data points. That is,

$$\nabla h_m(\mathbf{x}_i)^T\hat{\dot{\mathbf{x}}}_i = \mathbf{w}^T\mathrm{J}(\mathbf{x}_i)\hat{\dot{\mathbf{x}}}_i \geq 0 \quad \forall i \in \mathcal{I}_+ \tag{4}$$

where $\mathrm{J} \in \mathbb{R}^{F \times N}$ is the Jacobian matrix given by $\mathrm{J} = \left[\ \nabla\phi_1(\mathbf{x})\nabla\phi_2(\mathbf{x})\cdots\nabla\phi_F(\mathbf{x})\ \right]^T$ and $\hat{\dot{\mathbf{x}}}_i = \dot{\mathbf{x}}_i/\|\dot{\mathbf{x}}_i\|$ is the normalized velocity at the $i-th$ data point.

**Stability:** Lastly, the gradient of the modulation function must vanish at the attractor of the positive class $\mathbf{x}^*$. This constraint can be expressed as

$$\nabla h_m(\mathbf{x}^*)^T\mathbf{e}_i = \mathbf{w}^T\mathrm{J}(\mathbf{x}^*)\mathbf{e}_i = 0 \quad \forall i = 1...N \tag{5}$$

where the set of vectors $\{\mathbf{e}_i\}_{i=1\cdots N}$ is the canonical basis of $\mathbb{R}^N$.

## 3.1   Primal & Dual forms

As in the standard SVM [18], we optimize for maximal margin between the positive and negative class, subject to constraints 3-5 above. This can be formulated as:

$$\min_{\mathbf{w},\xi_i} \frac{1}{2}\|\mathbf{w}\|^2 + C\sum_{i\in\mathcal{I}_+}\xi_i \quad \textbf{subject to} \quad \left.\begin{array}{rll} y_i\left(\mathbf{w}^T\phi(\mathbf{x}_i) + b\right) & \geq 1 & \forall i = 1\cdots P \\ \mathbf{w}^T\mathrm{J}(\mathbf{x}_i)\hat{\dot{\mathbf{x}}}_i + \xi_i & > 0 & \forall i \in \mathcal{I}_+ \\ \xi_i & > 0 & \forall i \in \mathcal{I}_+ \\ \mathbf{w}^T\mathrm{J}(\mathbf{x}^*)\mathbf{e}_i & = 0 & \forall i = 1\cdots N \end{array}\right\}. \tag{6}$$

Here $\xi_i \in \mathbb{R}$ are slack variables that relax the Lyapunov constraint in Eq. 4. We retain these in our formulation to accommodate noise in the data representing the dynamics. $C \in \mathbb{R}_+$ is a penalty parameter for the slack variables. The Lagrangian for the above problem can be written as

$$\mathcal{L}(\mathbf{w}, b, \boldsymbol{\alpha}, \boldsymbol{\beta}, \boldsymbol{\gamma}) = \frac{1}{2}\|\mathbf{w}\|^2 + C\sum_{i\in\mathcal{I}_+}\xi_i - \sum_{i\in\mathcal{I}_+}\mu_i\xi_i - \sum_{i=1}^{P}\alpha_i\left(y_i(\mathbf{w}^T\phi(\mathbf{x}_i) + b) - 1\right)$$

$$- \sum_{i\in\mathcal{I}_+}\beta_i\left(\mathbf{w}^T\mathrm{J}(\mathbf{x}_i)\hat{\dot{\mathbf{x}}}_i + \xi_i\right) + \sum_{i=1}^{N}\gamma_i\mathbf{w}^T\mathrm{J}(\mathbf{x}^*)\mathbf{e}_i \tag{7}$$

where $\alpha_i, \beta_i, \mu_i, \gamma_i$ are the Lagrange multipliers with $\alpha_i, \beta_i, \mu_i \in \mathbb{R}_+$ and $\gamma_i \in \mathbb{R}$. Employing a similar analysis as in the standard SVM, it can be shown that the corresponding dual is given by the constrained quadratic program:

$$\min_{\boldsymbol{\alpha},\boldsymbol{\beta},\boldsymbol{\gamma}} \frac{1}{2}\begin{bmatrix}\boldsymbol{\alpha}^T\boldsymbol{\beta}^T\boldsymbol{\gamma}^T\end{bmatrix}\begin{bmatrix} \mathrm{K} & \mathrm{G} & -\mathrm{G}_* \\ \mathrm{G}^T & \mathrm{H} & -\mathrm{H}_* \\ -\mathrm{G}_*^T & -\mathrm{H}_*^T & \mathrm{H}_{**} \end{bmatrix}\begin{bmatrix}\boldsymbol{\alpha} \\ \boldsymbol{\beta} \\ \boldsymbol{\gamma}\end{bmatrix} - \boldsymbol{\alpha}^T\overline{\mathbf{1}} \quad \textbf{subject to} \quad \begin{array}{ll} 0 \leq \alpha_i & \forall i = 1...P \\ 0 \leq \beta_i \leq C & \forall i \in \mathcal{I}_+ \\ \sum_{i=1}^{P}\alpha_i y_i = 0 \end{array}$$

where $\overline{\mathbf{1}} \in \mathbb{R}^P$ is a vector with all entries equal to one. Let $k : \mathbb{R}^N \times \mathbb{R}^N \mapsto \mathbb{R}$ represents the kernel function such that $k(\mathbf{x}_1, \mathbf{x}_2) = \phi^T(\mathbf{x}_1)\phi(\mathbf{x}_2)$. The matrices $\mathrm{K} \in \mathbb{R}^{P \times P}, \mathrm{G} \in \mathbb{R}^{P \times |\mathcal{I}_+|}, \mathrm{G}_* \in \mathbb{R}^{P \times N}, \mathrm{H} \in \mathbb{R}^{|\mathcal{I}_+| \times |\mathcal{I}_+|}, \mathrm{H}_* \in \mathbb{R}^{|\mathcal{I}_+| \times N}, \mathrm{H}_{**} \in \mathbb{R}^{N \times N}$ can be expressed in terms of the kernel function and its first and second order derivatives:

$$
\left.
\begin{aligned}
(\mathrm{K})_{ij} &= y_i y_j k(\mathbf{x}_i, \mathbf{x}_j) & ; & & (\mathrm{H})_{ij} &= \hat{\mathbf{x}}_i^T \frac{\partial^2 k(\mathbf{x}_i, \mathbf{x}_j)}{\partial \mathbf{x}_i \partial \mathbf{x}_j} \hat{\mathbf{x}}_j \\
(\mathrm{G})_{ij} &= y_i \left( \frac{\partial k(\mathbf{x}_i, \mathbf{x}_j)}{\partial \mathbf{x}_j} \right)^T \hat{\mathbf{x}}_j & ; & & (\mathrm{H}_*)_{ij} &= \hat{\mathbf{x}}_i^T \frac{\partial^2 k(\mathbf{x}_i, \mathbf{x}^*)}{\partial \mathbf{x}_i \partial \mathbf{x}^*} \mathbf{e}_j \\
(\mathrm{G}_*)_{ij} &= y_i \left( \frac{\partial k(\mathbf{x}_i, \mathbf{x}^*)}{\partial \mathbf{x}^*} \right)^T \mathbf{e}_j & ; & & (\mathrm{H}_{**})_{ij} &= \mathbf{e}_i^T \frac{\partial^2 k(\mathbf{x}^*, \mathbf{x}^*)}{\partial \mathbf{x}^* \partial \mathbf{x}^*} \mathbf{e}_j
\end{aligned}
\right\} \tag{8}
$$

where $(.)_{ij}$ denotes the $i, j-th$ entry of the corresponding matrix. Due to space constraints, detailed development of the dual and proof of the above relations are given in appendices A and B of the supplement material.

Note that since the matrices K, H and $\mathrm{H}_{**}$ are symmetric, the overall Hessian matrix for the resulting quadratic program is also symmetric. However, unlike the standard SVM dual, it may not be positive definite resulting in multiple solutions to the above problem. In our implementation, we use the interior point solver *IPOPT* [19] to find a local optimum. We initialize the iterations using the $\alpha$ found by running first a standard SVM classification problem. All entries of $\beta$ and $\gamma$ are set to $0^3$. The solution to the above problem yields a modulation function (see Eq. A.11 for proof) given by

$$
h_m(\mathbf{x}) = \sum_{i=1}^P \alpha_i y_i k(\mathbf{x}, \mathbf{x}_i) + \sum_{i \in \mathcal{I}_+} \beta_i \hat{\mathbf{x}}_i^T \frac{\partial k(\mathbf{x}, \mathbf{x}_i)}{\partial \mathbf{x}_i} - \sum_{i=1}^N \gamma_i \mathbf{e}_i^T \frac{\partial k(\mathbf{x}, \mathbf{x}^*)}{\partial \mathbf{x}^*} + b \tag{9}
$$

which can be further expanded depending on the choice of kernel. Expansions for the Radial Basis Function (rbf) kernel are given in Appendix C.

The modulation function 9 learned using the A-SVM has noticeable similarities with the standard SVM classifier function. The first summation term is composed of the $\alpha$ support vectors ($\alpha$-SV) which act as support to the classification hyperplane. The second term entails a new class of support vectors that perform a linear combination of the normalized velocity $\hat{\mathbf{x}}_i$ at the training data points $\mathbf{x}_i$. These $\beta$ support vectors ($\beta$-SVs) collectively contribute to the fulfillment of the Lyapunov constraint by introducing a positive slope in the modulation function value along the directions $\hat{\mathbf{x}}_i$. Figure 4 shows the influence of a $\beta$-SV for the rbf kernel $k(\mathbf{x}_i, \mathbf{x}_j) = e^{1/2\sigma^2 \|\mathbf{x}_i - \mathbf{x}_j\|^2}$ with $\mathbf{x}_i$ at the origin

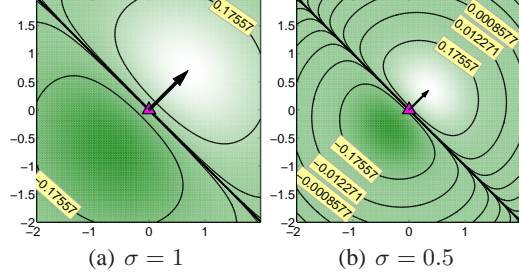

(a) $\sigma = 1$      (b) $\sigma = 0.5$

Figure 4: Isocurves of $f(\mathbf{x}) = \hat{\mathbf{x}}_i^T \frac{\partial k(\mathbf{x}, \mathbf{x}_i)}{\partial \mathbf{x}_i}$ at $\mathbf{x}_i = [0\ 0]^T, \hat{\mathbf{x}}_i = [\frac{1}{\sqrt{2}} \frac{1}{\sqrt{2}}]^T$ for the rbf kernel.

and $\hat{\mathbf{x}}_i = [\frac{1}{\sqrt{2}} \frac{1}{\sqrt{2}}]^T$. It can be seen that the smaller the kernel width $\sigma$, the steeper the slope. The third summation term is a non-linear bias, which does not depend on the chosen support vectors, and performs a local modification around the desired attractor $\mathbf{x}^*$ to ensure that the modulation function has a local maximum at that point. $b$ is the constant bias which normalizes the classification margins as $-1$ and $+1$. We calculate its value by making use of the fact that for all the $\alpha$-SV $\mathbf{x}_i$, we must have $y_i h_m(\mathbf{x}_i) = 1$. We use average of the values obtained from the different support vectors.

Figure 5 illustrates the effects of the support vectors in a 2D example by progressively adding them and overlaying the resulting DS flow in each case. The value of the modulation function $h_m(\mathbf{x})$ is shown by the color plot (white indicates high values). As the $\beta$-SVs are added in Figs. 5(b)-(d), they *push* the flow of trajectories along their associated directions. In Figs. 5(e)-(f), adding the two $\gamma$ terms shifts the location of the maximum of the modulation function to coincide with the desired attractor. Once all the SVs have been taken into account, the streamlines of the resulting DS achieve the desired criteria, i.e., they follow the training trajectories and terminate at the desired attractor.

[3]Source code for learning is available at http://asvm.epfl.ch

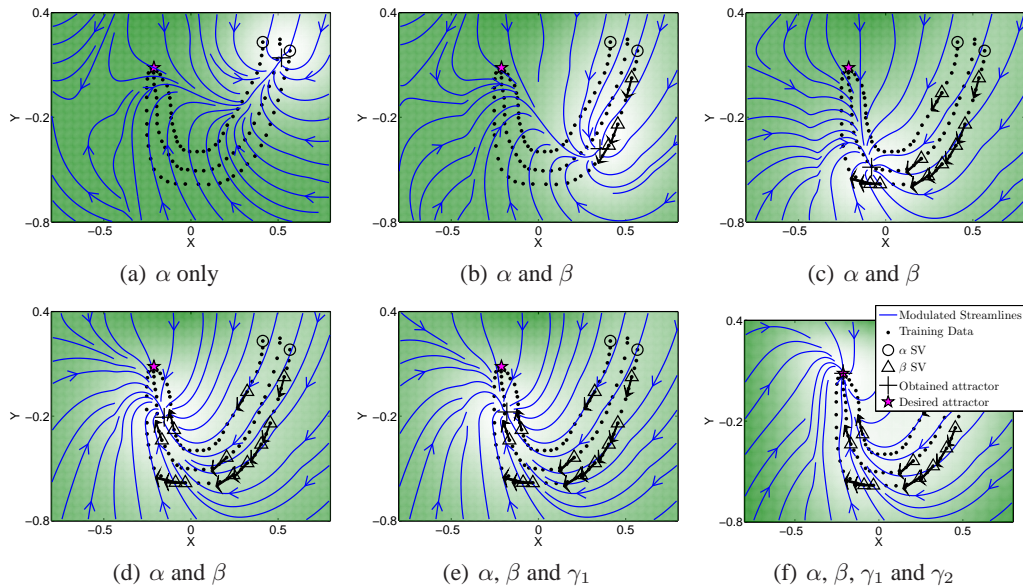

Figure 5: Progressively adding support vectors to highlight their effect on shaping the dynamics of the motion. (a) $\alpha$-SVs largely affect classification. (b)-(d) $\beta$-SVs guide the flow of trajectories along their respective associated directions $\hat{\mathbf{x}}_i$ shown by arrows. (e)-(f) The 2 $\gamma$ terms force the local maximum of the modulation function to coincide with the desired attractor along the $X$ and $Y$ axes respectively.

## 4   Results

In this section, we validate the presented A-SVM model on 2D (synthetic) data and on a robotic simulated experiment using a 7 degrees of freedom (DOF) KUKA-LWR arm mounted on a 3-DOF Omnirob base to catch falling objects. A video of the robotic experiment - simulated and real - is provided in Annexes. Next, we present a cross-validation analysis of the error introduced by the modulation in the original dynamics. A sensitivity analysis of the region of attraction of the resulting dynamical system with respect to the model parameters is also presented. We used the rbf kernel for all the results presented in this section. As discussed in Section 2, the RBF kernel is advantageous as it ensures that the function $h_m(\mathbf{x})$ is bounded. To generate an initial estimate of each individual dynamical system, we used the technique proposed in [7].

**2D Example** Figure 6(a) shows a synthetic example with 4 motion classes, each generated from a different closed form dynamics and containing 160 data points. The color plot indicates the value of the combined modulation function $\tilde{h}(\mathbf{x}) = \max_{m=1\cdots M} h_m(\mathbf{x})$ where each of the functions $h_m(\mathbf{x})$ are learned using the presented A-SVM technique. A total of 9 support vectors were obtained which is $< 10\%$ of the number of training data points. The trajectories obtained after modulating the original dynamical systems flow along increasing values of the modulation function, thereby bifurcating towards different attractors at the region boundaries. Unlike the dynamical system in Fig. 3, the flow here is aligned with the training trajectories and terminates at the desired attractors. To recall, this is made possible thanks to the additional constraints (Eq. 4 and 5) in our formulation.

In a second example, we tested the ability of our model to accommodate a higher density of attractors. We created 8 synthetic dynamics by capturing motion data using a screen mouse. Figure 1 shows the resulting 8 attractor system.

**Error Analysis** As formulated in Eq. 6, the Lyapunov constraints admit some slack, which allows the modulation to introduce slight deviations from the original dynamics. Here we statistically analyze this error via 5-fold cross validation. In the 4 attractor problem presented

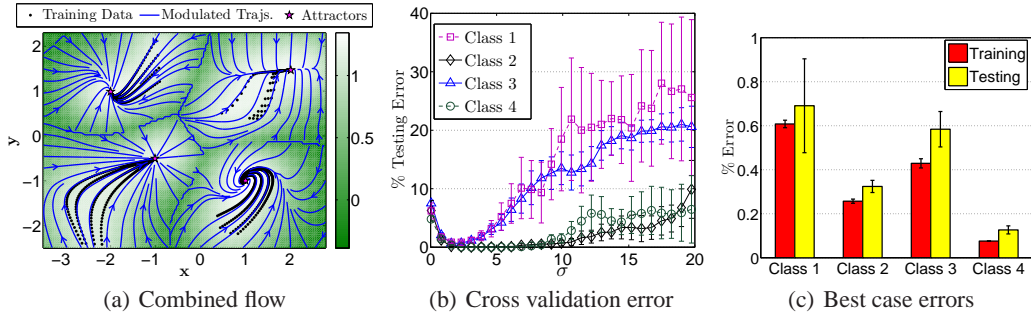

|     |     |     |
| --- | --- | --- |
| (a) Combined flow | (b) Cross validation error | (c) Best case errors |

Figure 6: Synthetic 2D case with 4-attractors.

above, we generate a total of 10 trajectories per motion class and use 2:3 training to testing ratio for cross validation. We calculate the average percentage error between the original velocity (read off from the data) and the modulated velocity (calculated using 2) for the $m - th$ class as $e_m = \left\langle \frac{\|\dot{\mathbf{x}}_i - \tilde{f}(\mathbf{x}_i)\|}{\|\dot{\mathbf{x}}_i\|} \times 100 \right\rangle_{i:l_i=m}$ where $< . >$ denotes average over the indicated range. Figure 6(b) shows the cross validation error (mean and standard deviation over the 5 folds) for a range of values of kernel width. The general trend revealed here is that for each class of motion, there exists a band of optimum values of the kernel width for which the testing error is the smallest. The region covered by this band of optimal values may vary depending on the relative location of the attractors and other data points. In Fig. 6(a), motion classes 2 (upper left) and 4 (upper right) are better fitted and show less sensitivity to the choice of kernel width than classes 1 (lower left) and 3 (lower right). We will show later in this section that this is correlated to the distance between the attractors. A comparison of testing and training errors for the least error case is shown in Fig. 6(c). We see that the testing errors for all the classes in the best case scenario are less than 1%.

**Sensitivity analysis** The partitioning of space created by our method results in $M$ regions of attraction (ROA) for each of our $M$ attractors. To assess the size of these regions and the existence of spurious attractors, we adopt an empirical approach. For each class, we compute the isosurfaces of the corresponding modulation function $h_m(\mathbf{x})$ in the range $[0, h_m(\mathbf{x}^*)]$. These hypersurfaces incrementally span the volume of the $m - th$ region around its attractor. We mesh each of these test surfaces and compute trajectories starting from the obtained mesh-points, looking for spurious attractors. $h_{ROA}$ is the isosurface of maximal value that encloses no spurious attractor and marks the ROA of the corresponding motion dynamics. We use the example in Fig. 5 to illustrate this process. Figure 7 shows a case where one spurious attractor is detected using a larger test surface (dotted line) whereas the actual ROA (solid line) is smaller. Once $h_{ROA}$ is calculated, we define the size of ROA as $r_{ROA} = (h(\mathbf{x}^*) - h_{ROA})/h(\mathbf{x}^*)$. $r_{ROA} = 0$

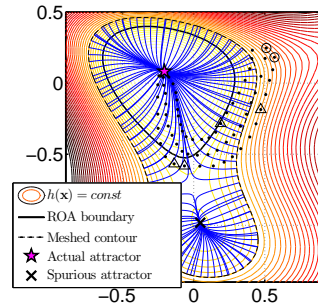

Figure 7: Test trajectories generated from several points on an isocurve (dotted line) to determine spurious attractors.

when no trajectory except those originating at the attractor itself, lead to the attractor. $r_{ROA} = 1$ when the ROA is bounded by the isosurface $h(\mathbf{x}) = 0$. The size of the $r_{ROA}$ is affected by both the choice of kernel width and the distance across nearby attractors. This is illustrated in Fig. 9 using data points from class 1 of Fig. 6(a) and translating the attractors so that they are either very far apart (left, distance $d_{att} = 1.0$) or very close to one another (right, $d_{att} = 0.2$). As expected, $r_{ROA}$ increases as we reach the optimal range of parameters. Furthermore, when the attractors are farther apart, high values of $r_{ROA}$ are obtained for a larger range of values of the kernel width, i.e., the model is less sensitive to the chosen kernel width. With smaller distance between the attractors (Fig. 9(b)), only a small deviation from the optimum kernel width results in a considerable loss in $r_{ROA}$, exhibiting high sensitivity to the model parameter.

**3D Example** We validated our method on a real world 3D problem. The attractors here represent manually labeled grasping points on a pitcher. The 3D model of the object was taken from the ROS IKEA object library. We use the 7-DOF KUKA-LWR arm mounted on the 3-DOF

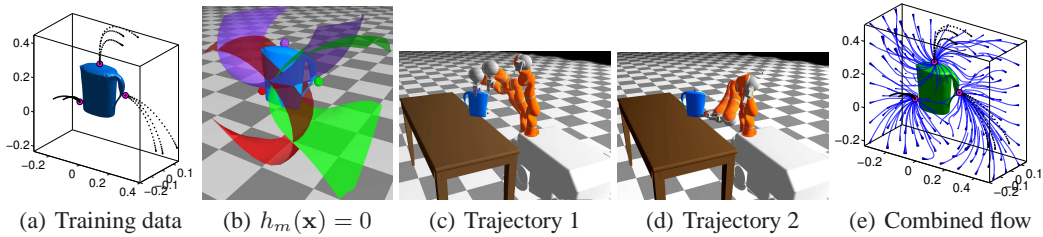

| (a) Training data | (b) $h_m(\mathbf{x}) = 0$ | (c) Trajectory 1 | (d) Trajectory 2 | (e) Combined flow |

Figure 8: 3D Experiment. (a) shows training trajectories for three manually chosen grasping points. (b) shows the isosurfaces $h_m(\mathbf{x}) = 0; m = 1, 2, 3$ along with the locations of the corresponding attractors. In (c) and (d), the robot executes the generated trajectories starting from different positions and hence converging to different grasping points. (e) shows the complete flow of motion.

KUKA-Omnirob base for executing the modulated Cartesian trajectories in simulation. We control all 10 DOF of the robot using the damped least square inverse kinematics. Training data for this implementation was obtained by recording the end-effector positions $\mathbf{x}_i \in \mathbb{R}^3$ from kinesthetic demonstrations of reach-to-grasp motions directed towards these grasping points, yielding a 3-class problem (see Fig. 8(a)). Each class was represented by 75 data points. Figure 8(b) shows the isosurfaces $h_m(\mathbf{x}) = 0; m \in \{1, 2, 3\}$ learned using the presented method. Figures 8(c)-(d) show the robot executing two trajectories when started from two different locations and converging to a different attractor (grasping point). Figure 8(e) shows the flow of motion around the object. Note that the time required to generate each trajectory point is $O(S)$ where $S$ denotes the total number of support vectors in the model. In this particular example with a total of 18 SVs, the trajectory points were generated at 1000 Hz which is well suited for real-time control. Such a fast generative model allows the robot to switch on-the-fly between the attractors and adapt to real-time perturbations in the object or the end-effector pose, without any re-planning or re-learning. Results for another object (champagne glass) are included in Appendix D (Fig. D.1). A video illustrating how the robot exploits multiple attractors to catch one of the grasping points on the object as it falls down is also provided in the supplementary material.

## 5    Conclusions

In this work, we presented the A-SVM model for combining non-linear dynamical systems through a partitioning of the space. We reformulated the optimization framework of SVM to encapsulate constraints that ensure accurate reproduction of the dynamics of motion. The new set of constraints result in a new class of support vectors that exploit partial derivatives of the kernel function to align the flow of trajectories with the training data. The resulting model

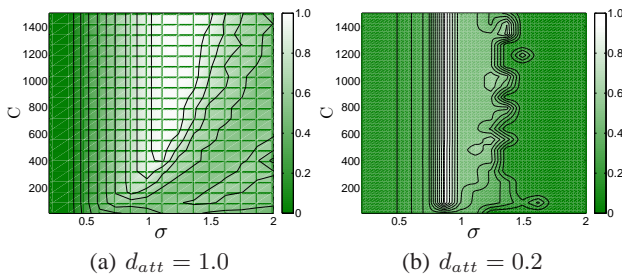

| (a) $d_{att} = 1.0$ | (b) $d_{att} = 0.2$ |

Figure 9: Variation of $r_{ROA}$ with varying model parameters.

behaves as a multi-stable DS with attractors at the desired locations. Each of the classified regions are forward invariant w.r.t the learned DS. This ensures that the trajectories do not cross over region boundaries. We validated the presented method on synthetic motions in 2D and 3D grasping motions on real objects. Results show that even though spurious attractors may occur, in practice they can be avoided by a careful choice of model parameters through grid search. The applicability of the method for real-time control of a 10-DOF robot was also demonstrated.

## Acknowledgments

This work was supported by EU Project *First-MM* (FP7/2007-2013) under grant agreement number *248258*. The authors would also like thank Prof. François Margot for his insightful comments on the technical material.

## Footnotes

[1]SVM classifier function is bounded if the Radial Basis Function (rbf) is used as kernel.

[2]Bold faced fonts represent vectors. $\mathbf{x}_i$ denotes the $i$-th vector and $x_i$ denotes the $i$-th element of vector $\mathbf{x}$.

# References

[1] Peter Pastor, Heiko Hoffmann, Tamim Asfour, and Stefan Schaal. Learning and generalization of motor skills by learning from demonstration. In *Robotics and Automation, 2009. ICRA '09. IEEE International Conference on*, pages 763 –768, may 2009.

[2] G. Schöner and M. Dose. A dynamical systems approach to task-level system integration used to plan and control autonomous vehicle motion. *Robotics and Autonomous Systems*, 10(4):253–267, 1992.

[3] G. Schöner, M. Dose, and C. Engels. Dynamics of behavior: Theory and applications for autonomous robot architectures. *Robotics and Autonomous Systems*, 16(2):213–245, 1995.

[4] L.P. Ellekilde and H.I. Christensen. Control of mobile manipulator using the dynamical systems approach. In *Robotics and Automation, 2009. ICRA'09. IEEE International Conference on*, pages 1370–1376. IEEE, 2009.

[5] H. Reimann, I. Iossifidis, and G. Schöner. Autonomous movement generation for manipulators with multiple simultaneous constraints using the attractor dynamics approach. In *Robotics and Automation (ICRA), 2011 IEEE International Conference on*, pages 5470–5477. IEEE, 2011.

[6] K.R. Dixon and P.K. Khosla. Trajectory representation using sequenced linear dynamical systems. In *Robotics and Automation, 2004. Proceedings. ICRA'04. 2004 IEEE International Conference on*, volume 4, pages 3925–3930. IEEE, 2004.

[7] S. M. Khansari-Zadeh and Aude Billard. Learning Stable Non-Linear Dynamical Systems with Gaussian Mixture Models. *IEEE Transaction on Robotics*, 2011.

[8] A. Shukla and A. Billard. Coupled dynamical system based armhand grasping model for learning fast adaptation strategies. *Robotics and Autonomous Systems*, 60(3):424 – 440, 2012.

[9] H. Hoffmann. Target switching in curved human arm movements is predicted by changing a single control parameter. *Experimental brain research*, 208(1):73–87, 2011.

[10] C. Rasmussen. Gaussian processes in machine learning. *Advanced Lectures on Machine Learning*, pages 63–71, 2004.

[11] S. Schaal, C.G. Atkeson, and S. Vijayakumar. Scalable techniques from nonparametric statistics for real time robot learning. *Applied Intelligence*, 17(1):49–60, 2002.

[12] Auke Jan Ijspeert, Jun Nakanishi, and Stefan Schaal. Movement imitation with nonlinear dynamical systems in humanoid robots. In *In IEEE International Conference on Robotics and Automation (ICRA2002*, pages 1398–1403, 2002.

[13] A. Fuchs and H. Haken. Pattern recognition and associative memory as dynamical processes in a synergetic system. i. translational invariance, selective attention, and decomposition ofscenes. *Biol. Cybern.*, 60:17–22, November 1988.

[14] A.N. Michel and J.A. Farrell. Associative memories via artificial neural networks. *Control Systems Magazine, IEEE*, 10(3):6 –17, apr 1990.

[15] H. Jaeger, M. Lukosevicius, D. Popovici, and U. Siewert. Optimization and applications of echo state networks with leaky-integrator neurons. *Neural Networks*, 20(3):335–352, 2007.

[16] J. Lee. Dynamic gradient approaches to compute the closest unstable equilibrium point for stability region estimate and their computational limitations. *Automatic Control, IEEE Transactions on*, 48(2):321–324, 2003.

[17] H.D. Chiang and C.C. Chu. A systematic search method for obtaining multiple local optimal solutions of nonlinear programming problems. *Circuits and Systems I: Fundamental Theory and Applications, IEEE Transactions on*, 43(2):99–109, 1996.

[18] B. Schölkopf and A.J. Smola. *Learning with kernels: Support vector machines, regularization, optimization, and beyond*. MIT press, 2001.

[19] Andreas Wchter and Lorenz T. Biegler. On the implementation of an interior-point filter line-search algorithm for large-scale nonlinear programming. *Mathematical Programming*, 106:25–57, 2006.

